# Learning Bayesian belief networks with neural network estimators

**Stefano Monti***
*Intelligent Systems Program
University of Pittsburgh
901M CL, Pittsburgh, PA – 15260
smonti@isp.pitt.edu

**Gregory F. Cooper*,****
**Center for Biomedical Informatics
University of Pittsburgh
8084 Forbes Tower, Pittsburgh, PA – 15261
gfc@cbmi.upmc.edu

## Abstract

In this paper we propose a method for learning Bayesian belief networks from data. The method uses artificial neural networks as probability estimators, thus avoiding the need for making prior assumptions on the nature of the probability distributions governing the relationships among the participating variables. This new method has the potential for being applied to domains containing both discrete and continuous variables arbitrarily distributed. We compare the learning performance of this new method with the performance of the method proposed by Cooper and Herskovits in [7]. The experimental results show that, although the learning scheme based on the use of ANN estimators is slower, the learning accuracy of the two methods is comparable.

Category: Algorithms and Architectures.

## 1 Introduction

Bayesian belief networks (BBN) are a powerful formalism for representing and reasoning under uncertainty. This representation has a solid theoretical foundation [13], and its practical value is suggested by the rapidly growing number of areas to which it is being applied. BBNs concisely represent the joint probability distribution over a set of random variables, by explicitly identifying the probabilistic dependencies and independencies between these variables. Their clear semantics make BBNs particularly suitable for being used in tasks such as diagnosis, planning, and control.

The task of learning a BBN from data can usually be formulated as a search over the space of network structures, and as the subsequent search for an optimal parametrization of the discovered structure or structures. The task can be further complicated by extending the search to account for hidden variables and for

the presence of data points with missing values. Different approaches have been successfully applied to the task of learning probabilistic networks from data [5]. In all these approaches, simplifying assumptions are made to circumvent practical problems in the implementation of the theory. One common assumption that is made is that all variables are discrete, or that all variables are continuous and normally distributed.

In this paper, we propose a novel method for learning BBNs from data that makes use of artificial neural networks (ANN) as probability distribution estimators, thus avoiding the need for making prior assumptions on the nature of the probability distribution governing the relationships among the participating variables. The use of ANNs as probability distribution estimators is not new [3], and its application to the task of learning Bayesian belief networks from data has been recently explored in [11]. However, in [11] the ANN estimators were used in the parametrization of the BBN structure only, and cross validation was the method of choice for comparing different network structures. In our approach, the ANN estimators are an essential component of the scoring metric used to search over the BBN structure space. We ran several simulations to compare the performance of this new method with the learning method described in [7]. The results show that, although the learning scheme based on the use of ANN estimators is slower, the learning accuracy of the two methods is comparable.

The rest of the paper is organized as follows. In Section 2 we briefly introduce the Bayesian belief network formalism and some basics of how to learn BBNs from data. In Section 3, we describe our learning method, and detail the use of artificial neural networks as probability distribution estimators. In Section 4 we present some experimental results comparing the performance of this new method with the one proposed in [7]. We conclude the paper with some suggestions for further research.

## 2 Background

A Bayesian belief network is defined by a triple $(G, \Omega, P)$, where $G = (\mathcal{X}, E)$ is a directed acyclic graph with a set of nodes $\mathcal{X} = \{x_1, \ldots, x_n\}$ representing domain variables, and with a set of arcs $E$ representing probabilistic dependencies among domain variables; $\Omega$ is the space of possible instantiations of the domain variables[1]; and $P$ is a probability distribution over the instantiations in $\Omega$. Given a node $x \in \mathcal{X}$, we use $\pi_x$ to denote the set of parents of $x$ in $\mathcal{X}$. The essential property of BBNs is summarized by the *Markov condition*, which asserts that each variable is independent of its non-descendants given its parents. This property allows for the representation of the multivariate joint probability distribution over $\mathcal{X}$ in terms of the univariate conditional distributions $P(x_i \mid \pi_i, \boldsymbol{\theta}_i)$ of each variable $x_i$ given its parents $\pi_i$, with $\boldsymbol{\theta}_i$ the set of parameters needed to fully characterize the conditional probability. Application of the chain rule, together with the Markov condition, yields the following factorization of the joint probability of any particular instantiation of all $n$ variables:

$$P(x'_1, \ldots, x'_n) = \prod_{i=1}^{n} P(x'_i \mid \pi'_{x_i}, \boldsymbol{\theta}_i) . \tag{1}$$

## 2.1   Learning Bayesian belief networks

The task of learning BBNs involves learning the network structure and learning the parameters of the conditional probability distributions. A well established set of learning methods is based on the definition of a scoring metric measuring the fitness of a network structure to the data, and on the search for high-scoring network structures based on the defined scoring metric [7, 10]. We focus on these methods, and in particular on the definition of Bayesian scoring metrics.

In a Bayesian framework, ideally classification and prediction would be performed by taking a weighted average over the inferences of every possible belief network containing the domain variables. Since this approach is in general computationally infeasible, often an attempt has been made to use a high scoring belief network for classification. We will assume this approach in the remainder of this paper.

The basic idea of the Bayesian approach is to maximize the probability $P(B_S \mid \mathcal{D}) = P(B_S, \mathcal{D})/P(\mathcal{D})$ of a network structure $B_S$ given a database of cases $\mathcal{D}$. Because for all network structures the term $P(\mathcal{D})$ is the same, for the purpose of model selection it suffices to calculate $P(B_S, \mathcal{D})$ for all $B_S$. The Bayesian metrics developed so far all rely on the following assumptions: 1) given a BBN structure, all cases in $\mathcal{D}$ are drawn independently from the same distribution (multinomial sample); 2) there are no cases with missing values (complete database); 3) the parameters of the conditional probability distribution of each variable are independent (*global parameter independence*); and 4) the parameters associated with each instantiation of the parents of a variable are independent (*local parameter independence*).

The application of these assumptions allows for the following factorization of the probability $P(B_S, \mathcal{D})$

$$P(B_S, \mathcal{D}) = P(B_S)P(\mathcal{D} \mid B_S) = P(B_S)\prod_{i=1}^{n} s(x_i, \pi_i, \mathcal{D}) , \qquad (2)$$

where $n$ is the number of nodes in the network, and each $s(x_i, \pi_i, \mathcal{D})$ is a term measuring the contribution of $x_i$ and its parents $\pi_i$ to the overall score of the network $B_S$. The exact form of the terms $s(x_i\, \pi_i, \mathcal{D})$ slightly differs in the Bayesian scoring metrics defined so far, and for lack of space we refer the interested reader to the relevant literature [7, 10].

By looking at Equation (2), it is clear that if we assume a uniform prior distribution over all network structures, the scoring metric is decomposable, in that it is just the product of the $s(x_i, \pi_i, \mathcal{D})$ over all $x_i$ times a constant $P(B_S)$. Suppose that two network structures $B_S$ and $B_{S'}$ differ only for the presence or absence of a given arc into $x_i$. To compare their metrics, it suffices to compute $s(x_i, \pi_i, \mathcal{D})$ for both structures, since the other terms are the same. Likewise, if we assume a decomposable prior distribution over network structures, of the form $P(B_S) = \prod_i p_i$, as suggested in [10], the scoring metric is still decomposable, since we can include each $p_i$ into the corresponding $s(x_i, \pi_i, \mathcal{D})$.

Once a scoring metric is defined, a search for a high-scoring network structure can be carried out. This search task (in several forms) has been shown to be NP-hard [4, 6]. Various heuristics have been proposed to find network structures with a high score. One such heuristic is known as K2 [7], and it implements a greedy search over the space of network structures. The algorithm assumes a given ordering on the variables. For simplicity, it also assumes that no prior information on the network is available, so the prior probability distribution over the network structures is uniform and can be ignored in comparing network structures.

The Bayesian scoring metrics developed so far either assume discrete variables [7, 10], or continuous variables normally distributed [9]. In the next section, we propose a possible generalization which allows for the inclusion of both discrete and continuous variables with arbitrary probability distributions.

## 3 An ANN-based scoring metric

The main idea of this work is to use artificial neural networks as probability estimators, to define a decomposable scoring metric for which no informative priors on the class, or classes, of the probability distributions of the participating variables are needed. The first three of the four assumptions described in the previous section are still needed, namely, the assumption of a multinomial sample, the assumption of a complete database, and the assumption of global parameter independence. However, the use of ANN estimators allows for the elimination of the assumption of local parameter independence. In fact, the conditional probabilities corresponding to the different instantiations of the parents of a variable are represented by the same ANN, and they share the same network weights and the same training data.

Let us denote with $\mathcal{D}_l \equiv \{C_1, \ldots, C_{l-1}\}$ the set of the first $l$ cases in the database, and with $x_i^{(l)}$ and $\pi_i^{(l)}$ the instantiations of $x_i$ and $\pi_i$ in the $l$-th case respectively. The joint probability $P(B_S, \mathcal{D})$ can be written as:

$$
P(B_{S_i}\mathcal{D}) = P(B_S)P(\mathcal{D}\,|\,B_S) = P(B_S)\prod_{l=1}^{m} P(C_l\,|\,\mathcal{D}_l, B_S) =
$$

$$
= P(B_S)\prod_{l=1}^{m}\prod_{i=1}^{n} P(x_i^{(l)}\,|\,\pi_i^{(l)}, \mathcal{D}_l, B_S). \tag{3}
$$

If we assume uninformative priors, or decomposable priors on network structures, of the form $P(B_S) = \prod_i p_i$, the probability $P(B_S, \mathcal{D})$ is decomposable. In fact, we can interchange the two products in Equation 3, so as to obtain

$$
P(B_S, \mathcal{D}) = \prod_{i=1}^{n}\Big[\,p_i \prod_{l=1}^{m} P(x_i^{(l)}\,|\,\pi_i^{(l)}, \mathcal{D}_l, B_S)\,\Big] = \prod_{i=1}^{n} s(x_i, \pi_i, \mathcal{D})\,, \tag{4}
$$

where $s(x_i, \pi_i, \mathcal{D})$ is the term between square brackets, and it is only a function of $x_i$ and its parents in the network structure $B_S$ ($p_i$ can be neglected if we assume a uniform prior over the network structures). The computation of Equation 4 corresponds to the application of the prequential method discussed by Dawid [8].

The estimation of each term $P(x_i\,|\,\pi_i, \mathcal{D}_l, B_S)$ can be done by means of neural network. Several schemes are available for training a neural network to approximate a given probability distribution, or density. Notice that the calculation of each term $s(x_i, \pi_i, \mathcal{D})$ can be computationally very expensive. For each node $x_i$, computing $s(x_i, \pi_i, \mathcal{D})$ requires the training of $m$ ANNs, where $m$ is the size of the database. To reduce this computational cost, we use the following approximation, which we call the $t$-invariance approximation: for any $l \in \{1, \ldots, m-1\}$, given the probability $P(x_i\,|\,\pi_i, \mathcal{D}_l, B_S)$, at least $t$ $(1 \le t \le m-l)$ new cases are needed in order to alter such probability. That is, for each positive integer $h$, such that $h < t$, we assume $P(x_i\,|\,\pi_i, \mathcal{D}_{l+h}, B_S) = P(x_i\,|\,\pi_i, \mathcal{D}_l, B_S)$. Intuitively, this approximation implies the assumption that, given our present belief about the value of each $P(x_i\,|\,\pi_i, \mathcal{D}_l, B_S)$, at least $t$ new cases are needed to revise this belief. By making this approximation, we achieve a $t$-fold reduction in the computation needed, since we now need to build and train only $m/t$ ANNs for each $x_i$, instead of the original $m$. In fact, application

of the $t$-invariance approximatioin to the computation of a given $s(x_i, \pi_i, \mathcal{D})$ yields:

$$s(x_i, \pi_i, \mathcal{D}) = \prod_{l=1}^{m} P(x_i^{(l)} \mid \pi_i^{(l)}, \mathcal{D}_l, B_S) = \prod_{k=0}^{m/t-1} \prod_{l=tk+1}^{t(k+1)} P(x_i^{(l)} \mid \pi_i^{(l)}, \mathcal{D}_{tk}, B_S). \,(5)$$

Rather than selecting a constant value for $t$, we can choose to increment $t$ as the size of the training database $\mathcal{D}_l$ increases. This approach seems preferable. When estimating $P(x_i \mid \pi_i, \mathcal{D}_l, B_S)$, this estimate will be very sensitive to the addition of new cases when $l$ is small, but will become increasingly insensitive to the addition of new cases as $l$ grows. A scheme for the incremental updating of $t$ can be summarized in the equation $t = \lceil \lambda l \rceil$, where $l$ is the number of cases already seen (i.e., the cardinality of $\mathcal{D}_l$), and $0 < \lambda \leq 1$. For example, given a data set of 50 cases, the updating scheme $t = \lceil 0.5l \rceil$ would require the training of the ANN estimators $P(x_i \mid \pi_i, \mathcal{D}_l, B_S)$ for $l = 1, 2, 3, 5, 8, 12, 18, 27, 41$.

## 4  Evaluation

In this section, we describe the experimental evaluation we conducted to test the feasibility of use of the ANN-based scoring metric developed in the previous section. All the experiments are performed on the belief network Alarm, a multiply-connected network originally developed to model anesthesiology problems that may occur during surgery [2]. It contains 37 nodes/variables and 46 arcs. The variables are all discrete, and take between 2 and 4 distinct values. The database used in the experiments was generated from Alarm, and it is the same database used in [7].

In the experiments, we use a modification of the algorithm K2 [7]. The modified algorithm, which we call ANN-K2, replaces the closed-form scoring metric developed in [7] with the ANN-based scoring metric of Equation (5). The performance of ANN-K2 is measured in terms of accuracy of the recovered network structure, by counting the number of edges added and omitted with respect to the Alarm network; and in terms of the accuracy of the learned joint probability distribution, by computing its cross entropy with respect to the joint probability distribution of Alarm. The learning performance of ANN-K2 is also compared with the performance of K2. To train the ANNs, we used the conjugate-gradient search algorithm [12].

Since all the variables in the Alarm network are discrete, the ANN estimators are defined based on the softmax model, with normalized exponential output units, and with cross entropy as cost function. As a regularization technique, we augment the training set so as to induce a uniform conditional probability over the unseen instantiations of the ANN input. Given the probability $P(x_i \mid \pi_i, \mathcal{D}_l)$ to be estimated, and assuming $x_i$ is a $k$-valued variable, for each instantiation $\pi_i'$ that does not appear in the database $D_l$, we generate $k$ new cases, with $\pi_i$ instantiated to $\pi_i'$, and $x_i$ taking each of its $k$ values. As a result, the neural network estimates $P(x_i \mid \pi_i', \mathcal{D}_l)$ to be uniform, with $P(x_i \mid \pi_i', \mathcal{D}_l) = 1/k$ for each of $x_i$'s values $x_{11}, \ldots, x_{1k}$.

We ran simulations where we varied the size of the training data set (100, 500, 1000, 2000, and 3000 cases), and the value of $\lambda$ in the updating scheme $t = \lceil \lambda l \rceil$ described in Section 3. We used the settings $\lambda = 0.35$, and $\lambda = 0.5$. For each run, we measured the number of arcs added, the number of arcs omitted, the cross entropy, and the computation time, for each variable in the network. That is, we considered each node, together with its parents, as a simple BBN, and collected the measures of interest for each of these BBNs. Table 1 reports mean and standard deviation of each measure over the 37 variables of Alarm, for both ANN-K2 and K2. The results for ANN-K2 shown in Table 1 correspond to the setting $\lambda = 0.5$,

| Data set | Algo. | arcs + | | arcs − | | cross entropy | | | time (secs) | | |
|---|---|---|---|---|---|---|---|---|---|---|---|
| | | *m* | *s.d.* | *m* | *s.d.* | *m* | *med* | *s.d.* | *m* | *med* | *s.d.* |
| 100 | ANN-K2 | 0.19 | 0.40 | 0.62 | 0.86 | 0.23 | .051 | 0.52 | 130 | 88 | 159 |
| | K2 | 0.75 | 1.28 | 0.22 | 0.48 | 0.08 | .070 | 0.10 | 0.44 | .06 | 1.48 |
| 500 | ANN-K2 | 0.19 | 0.40 | 0.22 | 0.48 | 0.04 | .010 | 0.11 | 1077 | 480 | 1312 |
| | K2 | 0.22 | 0.42 | 0.11 | 0.31 | 0.02 | .010 | 0.02 | 0.13 | .06 | 0.22 |
| 1000 | ANN-K2 | 0.24 | 0.49 | 0.22 | 0.48 | 0.05 | .005 | 0.15 | 6909 | 4866 | 6718 |
| | K2 | 0.11 | 0.31 | 0.03 | 0.16 | 0.01 | .006 | 0.01 | 0.34 | .23 | 0.46 |
| 2000 | ANN-K2 | 0.19 | 0.40 | 0.11 | 0.31 | 0.02 | .002 | 0.06 | 6458 | 4155 | 7864 |
| | K2 | 0.05 | 0.23 | 0.03 | 0.16 | 0.005 | .002 | 0.007 | 0.46 | .44 | 0.65 |
| 3000 | ANN-K2 | 0.16 | 0.37 | 0.05 | 0.23 | 0.01 | .001 | 0.017 | 11155 | 4672 | 2136 |
| | K2 | 0.00 | 0.00 | 0.03 | 0.16 | 0.004 | .001 | 0.005 | 1.02 | .84 | 1.11 |

Table 1: Comparison of the performance of ANN-K2 and of K2 in terms of number of arcs wrongly added (+), number of arcs wrongly omitted (−), cross entropy, and computation time. Each column reports the mean *m*, the median *med*, and the standard deviation *s.d.* of the corresponding measure over the 37 nodes/variables of Alarm. The median for the number of arcs added and omitted is always 0, and is not reported.

since their difference from the results corresponding to the setting $\lambda = 0.35$ was not statistically significant.

Standard t-tests were performed to assess the significance of the difference between the measures for K2 and the measures for ANN-K2, for each data set cardinality. No technique to correct for multiple-testing was applied. Most measures show no statistically significant difference, either at the 0.05 level or at the 0.01 level (most $p$ values are well above 0.2). In the simulation with 100 cases, both the difference between the mean number of arcs added and the difference between the mean number of arcs omitted are statistically significant ($p \simeq 0.01$). However, these differences cancel out, in that ANN-K2 adds fewer extra arcs than K2, and K2 omits fewer arcs than ANN-K2. This is reflected in the corresponding cross entropies, whose difference is not statistically significant ($p = 0.08$). In the simulation with 1000 cases, only the difference in the number of arcs omitted is statistically significant ($p \simeq .03$). Finally, in the simulation with 3000 cases, only the difference in the number of arcs added is statistically significant ($p \simeq .02$). K2 misses a single arc, and does not add any extra arc, and this is the best result to date. By comparison, ANN-K2 omits 2 arcs, and adds 5 extra arcs. For the simulation with 3000 cases, we also computed Wilcoxon rank sum tests. The results were consistent with the t-test results, showing a statistically significant difference only in the number of arcs added. Finally, as it can be noted in Table 1, the difference in computation time is of several order of magnitude, thus making a statistical analysis superfluous.

A natural question to ask is how sensitive is the learning procedure to the order of the cases in the training set. Ideally, the procedure would be insensitive to this order. Since we are using ANN estimators, however, which perform a greedy search in the solution space, particular permutations of the training cases might cause the ANN estimators to be more susceptible to getting stuck in local maxima. We performed some preliminary experiments to test the sensitivity of the learning procedure to the order of the cases in the training set. We ran few simulations in which we randomly changed the order of the cases. The recovered structure was identical in all simulations. Morevoer, the difference in cross entropy for different orderings of the cases in the training set showed not to be statistically significant.

## 5  Conclusions

In this paper we presented a novel method for learning BBNs from data based on the use of artificial neural networks as probability distribution estimators. As a prelim-

inary evaluation, we have compared the performance of the new algorithm with the performance of K2, a well established learning algorithm for discrete domains, for which extensive empirical evaluation is available [1, 7]. With regard to the learning accuracy of the new method, the results are encouraging, being comparable to state-of-the-art results for the chosen domain. The next step is the application of this method to domains where current techniques for learning BBNs from data are not applicable, namely domains with continuous variables not normally distributed, and domains with mixtures of continuous and discrete variables. The main drawback of the new algorithm is its time requirements. However, in this preliminary evaluation, our main concern was the learning accuracy of the algorithm, and little effort was spent in trying to optimize its time requirements. We believe there is ample room for improvement in the time performance of the algorithm. More importantly, the scoring metric of Section 3 provides a general framework for experimenting with different classes of probability estimators. In this paper we used ANN estimators, but more efficient estimators can easily be adopted, especially if we assume the availability of prior information on the class of probability distributions to be used.

## Acknowledgments

This work was funded by grant IRI-9509792 from the National Science Foundation.

## Footnotes

[1] An instantiation $\omega$ of all $n$ variables in $\mathcal{X}$ is an $n$-uple of values $\{x'_1, \ldots, x'_n\}$ such that $x_i = x'_i$ for $i = 1 \ldots n$.

## References

[1] C. Aliferis and G. F. Cooper. An evaluation of an algorithm for inductive learning of Bayesian belief networks using simulated data sets. In *Proceedings of the 10th Conference of Uncertainty in AI*, pages 8–14, San Francisco, California, 1994.

[2] I. Beinlich, H. Suermondt, H. Chavez, and G. Cooper. The ALARM monitoring system: A case study with two probabilistic inference techniques for belief networks. In *2nd Conference of AI in Medicine Europe*, pages 247–256, London, England, 1989.

[3] C. Bishop. *Neural Networks for Pattern Recognition*. Oxford University Press, 1995.

[4] R. Bouckaert. Properties of learning algorithms for Bayesian belief networks. In *Proceedings of the 10th Conference of Uncertainty in AI*, pages 102–109, 1994.

[5] W. Buntine. A guide to the literature on learning probabilistic networks from data. *IEEE Transactions on Knowledge and Data Engineering*, 1996. To appear.

[6] D. Chickering, D. Geiger, and D. Heckerman. Learning Bayesian networks: search methods and experimental results. *Proc. 5th Workshop on AI and Statistics*, 1995.

[7] G. Cooper and E. Herskovits. A Bayesian method for the induction of probabilistic networks from data. *Machine Learning*, 9:309–347, 1992.

[8] A. Dawid. Present position and potential developments: Some personal views. Statistical theory. The prequential approach. *Journal of Royal Statistical Society A*, 147:278–292, 1984.

[9] D. Geiger and D. Heckerman. Learning Gaussian networks. Technical Report MSR-TR-94-10, Microsoft Research, One Microsoft Way, Redmond, WA 98052, 1994.

[10] D. Heckerman, D. Geiger, and D. Chickering. Learning Bayesian networks: the combination of knowledge and statistical data. *Machine Learning*, 1995.

[11] R. Hofmann and V. Tresp. Discovering structure in continuous variables using Bayesian networks. In *Advances in NIPS 8*. MIT Press, 1995.

[12] M. Moller. A scaled conjugate gradient algorithm for fast supervised learning. *Neural Networks*, 6:525–533, 1993.

[13] J. Pearl. *Probabilistic Reasoning in Intelligent Systems: networks of plausible inference*. Morgan Kaufman Publishers, Inc., 1988.